# Adaptive Spatial Filters with predefined Region of Interest for EEG based Brain-Computer-Interfaces

**Moritz Grosse-Wentrup**
Institute of Automatic Control Engineering
Technische Universität München
80333 München, Germany
moritz@tum.de

**Klaus Gramann**
Department Psychology
Ludwig-Maximilians-Universität München
80802 München, Germany
gramann@psy.uni-muenchen.de

**Martin Buss**
Institute of Automatic Control Engineering
Technische Universität München
80333 München, Germany
mb@tum.de

## Abstract

The performance of EEG-based Brain-Computer-Interfaces (BCIs) critically depends on the extraction of features from the EEG carrying information relevant for the classification of different mental states. For BCIs employing imaginary movements of different limbs, the method of Common Spatial Patterns (CSP) has been shown to achieve excellent classification results. The CSP-algorithm however suffers from a lack of robustness, requiring training data without artifacts for good performance. To overcome this lack of robustness, we propose an adaptive spatial filter that replaces the training data in the CSP approach by a-priori information. More specifically, we design an adaptive spatial filter that maximizes the ratio of the variance of the electric field originating in a predefined region of interest (ROI) and the overall variance of the measured EEG. Since it is known that the component of the EEG used for discriminating imaginary movements originates in the motor cortex, we design two adaptive spatial filters with the ROIs centered in the hand areas of the left and right motor cortex. We then use these to classify EEG data recorded during imaginary movements of the right and left hand of three subjects, and show that the adaptive spatial filters outperform the CSP-algorithm, enabling classification rates of up to 94.7 % without artifact rejection.

## 1   Introduction

Brain-Computer-Interfaces (BCIs) allow communication without using the peripheral nervous systems by detecting intentional changes in the mental state of a user (see [1] for a review). For BCIs based on electroencephalography (EEG), different mental states are correlated with spatio-temporal pattern changes in the EEG. These can be detected and used for transmitting information by a suitable classification algorithm. While a variety of mental states can be used to induce pattern changes in the EEG, most BCIs utilize motor imagery of different limbs for this purpose. This is based on the observation that movement preparation of a certain limb leads to a power decrease (event related desynchronization - ERD) in the $\mu$- ($\sim 8 - 12$ Hz) and $\beta$-spectrum ($\sim 18 - 26$ Hz) over the area of the contralateral motor cortex representing the specific limb [2]. This ERD can also be observed in motor imagery, which was first used in [3] to discriminate imaginary movements of the left vs. imaginary movements of the right hand. While the methods presented in this paper are also applicable to BCIs that are not based on motor imagery, we restrict the discussion to this class of BCIs for

sake of simplicity.

In general, a BCI has to accomplish two tasks. The first task is the feature extraction, i.e., the extraction of information from the EEG relevant for discriminating different mental states. The second task is the actual classification of these feature vectors. For BCIs based on EEG, the feature extraction is aggravated by the fact that due to volume conduction only the superimposed electric activity of a large area of cortex can be measured at every electrode. While it is known that the ERD caused by motor imagery originates in the motor cortex [4], the EEG measured on the scalp over the motor cortex includes electrical activity of multiple neural sources that are not related to the imaginary movements. This in turn leads to a lower signal-to-noise-ratio (SNR) and subsequently to a lower classification accuracy. For this reason, algorithms have been developed that use multiple recording sites (electrodes) to improve feature extraction. One of the most successful algorithms in this context, as evidenced by the 2003 BCI Competition [5], is the method of Common Spatial Patterns (CSP). It was introduced to EEG analysis in [6] and first utilized for BCIs in [7]. Given two EEG data sets recorded during motor imagery of the left and right hand, the CSP-algorithm finds two linear transformations that maximize the variance of the one while minimizing the variance of the other data set. With the CSP-algorithm used for feature extraction, it has been shown that a simple linear classification algorithm suffices to obtain classification rates above 90 % for trained subjects [7]. While these are impressive results, the CSP algorithm suffers from a lack of robustness. The algorithm is trained to maximize the differences between two datasets, regardless of the cause of these differences. In the ideal case the spatio-temporal differences are caused only by the motor imagery, in which case the algorithm can be claimed to be optimal. In practice, however, the differences between two datasets will be due to multiple causes such as spontaneous EEG activity, other mental states or any kind of artifacts. For example, if a strong artifact is present in only one of the data sets, the CSP is trained on the artifact and not on the differences caused by motor imagery [7]. Consequently, the CSP algorithm requires artifact free data, which is a serious impairment for its practical use.

In this paper, we propose to replace the information used for training in the CSP algorithm by a-priori information. This is possible, since it is known that the signal of interest for the classification of imaginary movements originates in the motor cortex [4]. More specifically, we design an adaptive spatial filter (ASF) that maximizes the ratio of the variance of the electric field originating in a predefined region of interest (ROI) in the cortex and the overall variance of the EEG measurements. In this way, we can design spatial filters that optimally suppress electric activity originating from other areas than the chosen ROI. By designing two spatial filters with the respective ROIs centered in the hand areas of the motor cortex in the left and right hemisphere, we achieve a robust feature extraction that enables better classification results than obtained with the CSP-algorithm.

The rest of this paper is organized as follows. In the methods section, we briefly review the CSP-algorithm, derive the ASF and discuss its properties. In the results section, the ASF is applied to EEG data of three subjects, recorded during imaginary movements of the right and left hand. The results are compared with the CSP-algorithm, and it is shown that the ASF is superior to the CSP approach. This is evidenced by a significant increase in measured ERD and higher classification accuracy. We conclude the paper with a discussion of the results and future lines of research.

## 2 Methods

In this section we will first briefly review the CSP-algorithm, and then show how the information used for training the CSP-algorithm can be replaced by a-priori information. We then derive the ASF, and conclude the section with some remarks on the theoretical properties of the ASF.

### 2.1 The Common Spatial Patterns Algorithm

Given two EEG data sets $\boldsymbol{x}_1 \in \mathbb{R}^{N \times T}$ and $\boldsymbol{x}_2 \in \mathbb{R}^{N \times T}$ with $N$ the number of electrodes and $T$ the number of samples, the CSP-algorithm finds a linear transformation $\boldsymbol{w}$ that maximizes the variance of the one while minimizing the variance of the other data set. This can be formulated as the following optimization problem [8]:

$$\max_{\boldsymbol{w}} \left\{ \frac{\boldsymbol{w}^{\mathrm{T}} R_1 \boldsymbol{w}}{\boldsymbol{w}^{\mathrm{T}} R_2 \boldsymbol{w}} \right\} \tag{1}$$

with $R_1$ and $R_2$ the spatial covariance matrices of $\boldsymbol{x}_1$ and $\boldsymbol{x_2}$. This optimization problem is in the form of the well known Rayleigh quotient, which means that the solution to (1) is given by the eigenvector with the largest eigenvalue of the generalized eigenvalue problem

$$R_1\boldsymbol{w} = \lambda R_2\boldsymbol{w}. \tag{2}$$

The two eigenvectors $\boldsymbol{w}_1$ and $\boldsymbol{w}_2$ with the largest and smallest respective eigenvalue represent two spatial filters that maximize the variance of the one while minimizing the variance of the other data set. If data set $\boldsymbol{x}_1$ was recorded during imaginary movements of the left while data set $\boldsymbol{x}_2$ was recorded during imaginary movements of the right hand, and the differences between the two data sets are only caused by motor imagery, the two spatial filters $\boldsymbol{w}_1$ and $\boldsymbol{w}_2$ optimally (in terms of the second moments) extract the component of a data set caused by the respective motor imagery. In practice, however, the differences between two datasets will have multiple causes such as spontaneous EEG activity, mental states unrelated to the motor imagery or muscular artifacts. If the CSP-algorithm is applied to such data, the linear transformations $\boldsymbol{w}_{1/2}$ will be trained to extract the artifactual components of the EEG and not the spatial-temporal pattern changes caused by the motor imagery.

## 2.2 Derivation of the Adaptive Spatial Filter

To overcome the sensitivity of the CSP-algorithm to artifacts in the EEG, and achieve a robust feature extraction, we replace the information used for training in the CSP-algorithm by a-priori information. In the case of motor imagery, the specific a-priori information is that the signal of interest for classification originates in the motor cortex. We will now show how the component of the EEG originating in a certain ROI, chosen to correspond to the motor cortex for our purpose, can be estimated in an optimal manner.

In general, it would be desirable to derive a spatial filter that eliminates all electric activity that does not originate in a chosen ROI. This however is not possible due to the ill-posed nature of the inverse problem of EEG (c.f., [9]). In EEG recordings, electric activity originating from an infinite dimensional space (the continuous current distribution within the brain) is mapped onto a finite number of measurement electrodes. Hence, the best one can do is to find a spatial filter that in some sense optimally suppresses all activity not originating in the chosen ROI.

Towards this goal, note that the electric field generated by the brain at a position $\boldsymbol{r}$ outside the head is given by (c.f., [10])

$$\Phi(\boldsymbol{r},t) = \int_V L(\boldsymbol{r},\boldsymbol{r}')^{\mathrm{T}} P(\boldsymbol{r}',t)\mathrm{d}V(\boldsymbol{r}'), \tag{3}$$

with $V$ the volume of the brain, $P : \mathbb{R}^3 \times \mathbb{R} \mapsto \mathbb{R}^3$ the tissue dipole moment (source strength) at position $\boldsymbol{r}'$ and time $t$ in x, y, and z - direction, and $L : \mathbb{R}^3 \times \mathbb{R}^3 \mapsto \mathbb{R}^3$ the so called leadfield equation, describing the projection strength of a source with dipole moment in x, y, and z - direction at position $\boldsymbol{r}'$ to a measured electric field at position $\boldsymbol{r}$. Note that the leadfield equation incorporates all geometric and conductive properties of the brain. In EEG recordings, the electric field of the brain is spatially sampled at $i = 1\ldots N$ electrodes on the scalp with position $\boldsymbol{r}_i$, resulting in a measurement vector $\boldsymbol{x}(t)$ with the elements

$$x_i(t) = \int_V L(\boldsymbol{r}_i,\boldsymbol{r}')^{\mathrm{T}} P(\boldsymbol{r}',t)\mathrm{d}V(\boldsymbol{r}'), \ i = 1\ldots N. \tag{4}$$

We now wish to find a linear transformation of the measured EEG

$$\boldsymbol{y}(t) = \boldsymbol{w}^{\mathrm{T}}\boldsymbol{x}(t) \tag{5}$$

that maximizes the ratio of the variance of the electric field originating in a certain area of the cortex and the overall variance. For this we define the component of the EEG originating in a certain ROI as $\tilde{\boldsymbol{x}}(t)$, with the elements

$$\tilde{x}_i(t) = \int_{\mathrm{ROI}} L(\boldsymbol{r}_i,\boldsymbol{r}')^{\mathrm{T}} P(\boldsymbol{r}',t)\mathrm{d}V(\boldsymbol{r}'), \ i = 1\ldots N. \tag{6}$$

The spatial filter $\boldsymbol{w}$ is then found by

$$\max_{\boldsymbol{w}} \{f(\boldsymbol{w})\} \text{ with } f(\boldsymbol{w}) = \frac{\boldsymbol{w}^{\mathrm{T}}\tilde{\boldsymbol{x}}(t)\tilde{\boldsymbol{x}}(t)^{\mathrm{T}}\boldsymbol{w}}{\boldsymbol{w}^{\mathrm{T}}\boldsymbol{x}(t)\boldsymbol{x}(t)^{\mathrm{T}}\boldsymbol{w}} = \frac{\boldsymbol{w}^{\mathrm{T}}R_{\tilde{\boldsymbol{x}}}(t)\boldsymbol{w}}{\boldsymbol{w}^{\mathrm{T}}R_{\boldsymbol{x}}(t)\boldsymbol{w}} \tag{7}$$

and $R_{\tilde{x}}(t)$ and $R_{x}(t)$ the (spatial) covariance matrices of $\tilde{x}(t)$ and $x(t)$. Note that this optimization problem is in the same form as that of the CSP-algorithm in (1). As for (1), the solution to (7) is given by the eigenvector with the largest eigenvalue of the generalized eigenvalue problem

$$R_{\tilde{x}}(t)\boldsymbol{w} = \lambda R_{x}(t)\boldsymbol{w}. \tag{8}$$

*The crucial difference between the CSP- and the ASF-algorithm is that for the CSP-algorithm the covariance matrix $R_1$ in the numerator of (1), describing the signal subspace of the data, is given by the measured EEG of one condition. For the ASF-algorithm, the corresponding covariance matrix $R_{\tilde{x}}(t)$ is replaced by a-priori knowledge independent of the measured EEG.*

We will now show how estimates of the two covariance matrices $R_{\tilde{x}}(t)$ and $R_{x}(t)$, necessary for solving (8), can be obtained. Assuming stationarity of the EEG, i.e., a constant covariance matrix, $R_{x}(t)$ can be replaced by the estimated sample covariance matrix

$$\hat{R}_{\boldsymbol{x}} = \frac{1}{T} \sum_{t=1}^{T} \boldsymbol{x}(t)\boldsymbol{x}(t)^{\mathrm{T}}, \tag{9}$$

with $T$ the number of samples. The covariance matrix of the EEG originating in the ROI however is substantially harder to estimate. To obtain an estimate of $R_{\tilde{x}}(t)$, we first derive an estimate of (6). This is done by placing an equally spaced grid with nodes at locations $\boldsymbol{r}'_i$, $i = 1 \ldots M$ in the ROI, and replacing the integral over the ROI by a sum over the $M$ grid points,

$$\tilde{x}_i(t) = \sum_{j=1}^{M} L(\boldsymbol{r}'_j, \boldsymbol{r}_i)^{\mathrm{T}} P(\boldsymbol{r}'_j, t). \tag{10}$$

The estimated component of the EEG originating in the ROI can then be written in matrix notation as

$$\tilde{\boldsymbol{x}}(t) = L\boldsymbol{p}(t), \tag{11}$$

with $L \in \mathbb{R}^{N \times 3M}$ describing the projection strength of the $M$ sources in x, y, and z - direction to each of the $N$ electrodes, and $\boldsymbol{p}(t) \in \mathbb{R}^{3M}$ representing the dipole moments of the $M$ sources. The estimate of the covariance matrix is then given by

$$\hat{R}_{\tilde{\boldsymbol{x}}}(t) = L\boldsymbol{p}(t)\boldsymbol{p}(t)^{\mathrm{T}}L^{\mathrm{T}} = LR_{\boldsymbol{p}}(t)L^{\mathrm{T}}. \tag{12}$$

In absence of any prior knowledge, the covariance matrix of the sources in the ROI is assumed to be the identity matrix, i.e., $R_{\boldsymbol{p}}(t) = I_{3M}$. The leadfield matrix $L$ on the other hand can be estimated by a suitable model of EEG volume conduction. For sake of simplicity, we only consider a four-shell spherical head model, i.e., each column $l_i$ of the leadfield matrix $L$ is found by placing a single current dipole with unit dipole moment at position $r_i$ in a four shell spherical head model, and calculating its projection to each of the $N$ electrodes [11].

In summary, the adaptive linear spatial filter $\boldsymbol{w}$ is given by the eigenvector with the largest eigenvalue of the generalized eigenvalue problem

$$LL^{\mathrm{T}}\boldsymbol{w} = \lambda \hat{R}_{\boldsymbol{x}}\boldsymbol{w}. \tag{13}$$

Note that the largest eigenvalue corresponds to the achieved ratio of the ASF, i.e., $f(\boldsymbol{w}) = \lambda$. The largest eigenvalue of (13) thus is a measure for the quality of the obtained ASF.

It is also important to point out that the covariance matrix of the component of the EEG originating in the ROI is assumed to be the identity matrix, implying that sources in the ROI are not correlated. This surely is an assumption that is not physiologically justified. We will address this issue in the discussion. Finally, note that the quality of the obtained filter also depends on the rank of the covariance matrix of the electric activity originating in the ROI. The higher the rank of $R_{\tilde{x}}$, the more degrees of freedom of the spatial filter are required to pass activity from the signal subspace, i.e., activity originating in the ROI, and consequently less degrees of freedom are available for suppressing electric activity originating outside the ROI. The quality of the spatial filter thus decreases with the rank of $R_{\tilde{x}}$. For this reason, it is beneficial to only consider radially oriented dipole sources in the ROI, which leads to a covariance matrix with a lower rank than if dipole moments in x, y, and z - direction are considered. Furthermore, this is a physiologically justified assumption, since neurons in a cortical column are oriented radially to the surface of the cortex (c.f., [9]).

# 3  Results

In this section, we evaluate the effectiveness of the ASF by applying it to EEG data gathered from three subjects during motor imagery of the right and left hand, and compare its performance with the CSP-algorithm. Subsequently motor imagery of the left/right hand will be termed condition IL/IR.

## 3.1  Experimental Setup

Three subjects (S1, S2, S3) participated in the experiment, all of which were male, aged 26, 30, and 27 years, and had no known neurological disorders. Subjects S1 and S2 had no experience with motor imagery or BCIs, while subject S3 participated in a motor imagery experiment for the second time. The subjects were placed in a shielded room approximately two meters in front of a screen, and were asked to continually imagine opening and closing their right/left hand as long as an arrow pointing in the respective direction was displayed on the screen. The subjects were explicitly instructed to perform haptic motor imagery, i.e., to feel how they were opening and closing their hands, to ensure that actual motor imagery and not visual imagery was used. Each trial started with a fixation cross, which was superimposed by an arrow either pointing to the right or to the left after three seconds. The center of the arrow was placed in the middle of the screen to avoid lateralized visual evoked potentials. The arrow was removed again after further seven seconds, indicating the end of one trial. A total of 300 trials were recorded for each subject, consisting of 150 trials for each condition in randomized order. During the experiment, EEG was recorded at 128 channels with a sampling rate of 500 Hz. Electrode Cz was used as a reference, and the data was re-referenced offline to common average. The spatial position of each electrode was measured with a tracking system. No trials were rejected and no artifact correction was employed.

## 3.2  Design of the Common Spatial Patterns

For each subject, the CSPs were found by by first bandpass-filtering the data between 10 - 30 Hz (as suggested in [7]) using a sixth order butterworth filter, and then calculating the sample covariance matrices $R_1$ and $R_2$ for both conditions (IL and IR) of all trials in a time window ranging from 3.5 to 10 s (i.e., starting 500 ms after the instruction which motor imagery to perform). Equation (2) was then solved and only the two most discriminative eigenvectors $w_{1,\mathrm{CSP}}$ and $w_{2,\mathrm{CSP}}$, i.e., those with the largest and smallest respective eigenvalue, were used to obtain estimates of the two most discriminative components of each data set.

## 3.3  Design of the Adaptive Spatial Filters

To obtain estimates of the electric field originating in the hand areas of the left and right motor cortex, two ASFs were designed. For the first ASF, the ROI was chosen as a sphere located 1 cm inside the cortex radially below electrode C3 (centered above the hand area of the motor cortex of the left hemisphere) with a radius of 1 cm. Radially oriented dipoles with unit moment were placed on an equally spaced grid 2 mm apart from each other inside the sphere, and their respective projections to each of the electrodes were calculated as in [11] to obtain an estimate of the leadfield matrix $L$ in (11). For this purpose, the measured positions of the electrodes were radially projected onto the outermost sphere of the headmodel. The second ASF was designed in the same fashion, but with the center of the sphere located at the same depth as the first one radially below electrode C4 (centered above the hand area of the motor cortex of the right hemisphere).

For each of the 300 trials of each subject, the data covariance matrix of the recorded EEG was estimated according to (9), using EEG data in the same time window as for computation of the CSPS (3.5 to 10 s of each trial). The two ASFs with the ROIs centered below electrodes C3 and C4 were then calculated by solving the generalized eigenvalue problem (13), and taking the eigenvector with the largest eigenvalue as the ASF. The estimated activity inside the ROIs was then obtained by multiplying the ASFs with the observed EEG of that trial according to (5). Note that this was done independently for each recorded trial.

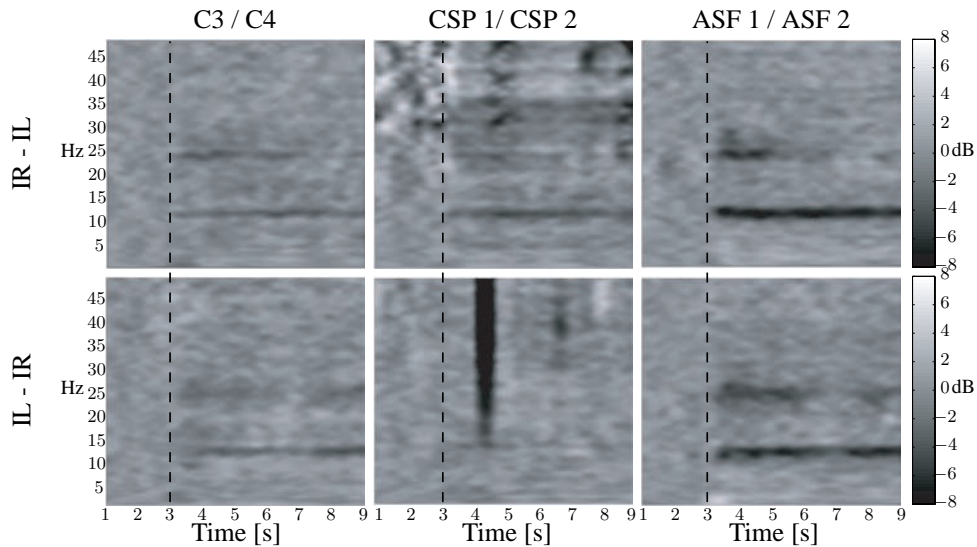

Figure 1: Difference plots of ERS/ERD relative to pre-stimulus baseline (0 - 3 s) between conditions IL and IR for subject S3. Time of stimulus onset is marked by the dotted vertical line. See text for explanations.

## 3.4 Experimental Results

To obtain estimates of the frequency bands suitable for classification, event related synchronization and desynchronization (ERS/ERD) was calculated for each subject relative to the pre-stimulus baseline (0 - 3 s) as implemented in [12]. This was done independently for the EEG measured at electrodes C3 and C4, the EEG components obtained by the CSP-algorithm and the estimated EEG components originating in the motor cortex as obtained by the ASFs. Since motor imagery leads to a contralateral ERD, the ERS/ERD of condition IL was subtracted from the ERS/ERD of condition IR for measurements at electrode C3 and spatial filters focusing on the left hemisphere, and vice versa for measurements at electrode C4 and spatial filters focusing on the right hemisphere. The results for subject S3 are shown in Fig. 1. As can be seen in the first column, an ERD-difference of about 3 dB can be measured over the contralateral motor cortex at electrodes C3 and C4 in two frequency bands, starting briefly after the instruction to perform the motor imagery. The two reactive frequency bands are centered roughly around 12 and 25 Hz, agreeing well with the expected ERD in the $\mu$- and $\beta$-band [3]. The components extracted by the CSPs, shown in the second column of Fig. 1, show a different picture. While the first CSP (top row) extracts the ERD in the $\mu$- and $\beta$-band with roughly the same SNR as in the first row of Fig. 1, high-frequency noise is also mixed in. The second CSP (bottom row) on the other hand does not extract the ERD related to motor imagery, but focuses on a strong artifactual component above 15 Hz. The third column of Fig. 1 shows the results obtained with the ASFs centered radially below electrodes C3 and C4. The observed ERD is similar to the one measured directly at electrodes C3 and C4 (first column of Fig. 1), but shows a much stronger ERD of about 7 dB in the $\mu$- and $\beta$-band. These observation are also reflected in the actual CSPs (calculated for the whole data set) and ASFs (calculated for one representative trial) of subject S3 shown in Fig. 2. While CSP 1 focuses on electrodes in the vicinity of electrode C3, CSP 2 focuses on frontal areas of the recorded EEG that are not related motor control. The ASFs on the other hand can be seen to focus on motor areas surrounding electrodes C3 and C4, with various minor patches distributed over the scalp that suppress electric activity originating outside the ROI. It is thus evident that the ASFs improve the SNR of the component of the EEG related to motor imagery relative to just measuring the ERD above the motor cortex, while the CSPs fail to extract the ERD related to motor imagery due to artifactual components. Similar ERS/ERD results were obtained for subject S2, while only a very weak ERS/ERD could be observed for subject S1 for all three evaluation schemes.

 The ERD plots at electrodes C3 and C4 of each subject were then used to heuristically determine the time window and two reactive frequency bands used for actual classification. These are summarized in Tab. 1. For classification, the reactive frequency bands of the data sets obtained by the

| CSP 1 | CSP 2 | ASF 1 | ASF 2 |

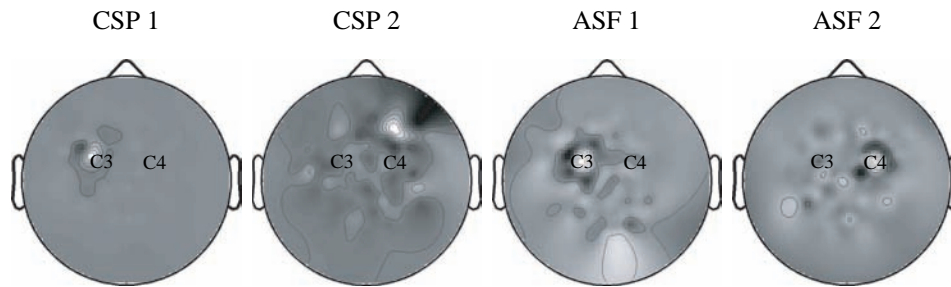

Figure 2: Spatial filters obtained by the CSP- and the ASF-algorithm for subject S3.

Table 1: Classification results

| SUBJECT | TIME WINDOW | FREQ. BANDS | C3/C4 | CSP | ASF |
|---------|-------------|-------------|-------|-----|-----|
| S1 | 3.5 - 10 s | 17 - 18 & 26 - 28 Hz | 58.0 % | 48.3 % | **63.0** % |
| S2 | 3.5 - 10 s | 9 - 12.5 & 23 - 26 Hz | 87.0 % | 49.0 % | **90.3** % |
| S3 | 3.5 - 10 s | 12 - 14 & 20 - 30 Hz | 77.3 % | 60.7 % | **94.7** % |

CSP-algorithm, the ASFs, and the raw EEG data measured at electrodes C3 and C4 were extracted by using a sixth-order butterworth filter. For each of the three evaluation approaches, the feature vectors were formed by calculating the variance in each of the two frequency bands for each trial. This resulted in a four dimensional feature vector for each trial and each evaluation approach. The feature vectors were then classified using leave-one-out cross validation with Fisher Linear Discriminant Analysis (c.f., [13]). Note that for the CSP algorithm this required recalculation of the CSPs for each cross validation. The classification results for each approach and all three subject are shown in Tab. 1.

As can be seen in Tab. 1, the ASFs lead to an increase in classification accuracy relative to measuring the ERS/ERD at electrodes C3 and C4 of 3.3 - 17.4 %. The CSP-algorithm on the other hand leads to worse classification results compared to only using the ERS/ERD measured at electrodes C3/C4. In fact, for S1 and S2 the classification accuracy was not above chance for the CSP-algorithm. In agreement with [7], the ERS/ERD as well as the CSP plots for subjects S1 and S2 (not shown here) indicate that this is due to the fact that the CSPs focus on artifactual components that are not related to motor imagery. Subjects S2 and S3 achieved a classification accuracy of 90.3 and 94.7 % using the ASFs, while subject S1 only achieved 63 %. This correlates with the personal report of the subjects, with S2 and S3 reporting that they considered their motor imagery to be successful, while S1 reported difficulties in imaging opening and closing his hands.

## 4 Discussion

In this paper, we presented a new approach for feature extraction for EEG-based BCIs. We derived an adaptive spatial filter that maximizes the ratio of the variance of the electric field originating in a specified ROI of cortex and the overall variance of the measured EEG. By designing two ASFs with the ROIs centered in the hand areas of the motor cortex, we showed that classification accuracy of imaginary movements of the left and right hand increased between 3.3 and 17.4 % relative to using the EEG measured directly above motor areas at electrodes C3 and C4. This was achieved without any artifact correction or rejection of trials. In contrast, applying the CSP-algorithm to the same data sets lead to a classification accuracy below that of only using recordings from electrodes C3 and C4 for feature extraction for one subject, and a classification accuracy that was not above chance for the other two subjects. This was due to the lack of robustness of the CSP-algorithm, focusing on artifactual components of the EEG. We thus conclude that the proposed ASF-method enables a significant increase in classification accuracy, and is very robust to artifactual components in the EEG.

While the presented results are already promising, several aspects of the ASF can be further optimized. These include the four-shell spherical head model used for estimating the leadfield matrix, which is the most simple and inaccurate model available in the literature. Employing more realistic

models for volume conduction, such as finite element or boundary element methods (FEM/BEM), are expected to further increase classification accuracy. Furthermore, the ROIs were heuristically chosen as spheres located radially below electrodes C3 and C4. Due to individual differences in physiology and/or misplacement of the electrode caps, the ROIs were unlikely to be centered in the hand areas of the motor cortex of each subject. Optimization of the center and extent of the ROI, either by a-priori knowledge gained by fMRI scans or by numerical optimization of the ERD in specific frequency bands, is expected to lead to higher SNRs and hence higher classification accuracy. Another issue that can be addressed to improve performance of the ASF is the physiologically not justified assumption of uncorrelated sources in the ROI.

Besides optimization issues of parameters, future lines of research include extending the algorithm to multi class problems, e.g., BCIs using motor imagery of more than two limbs. Conceptually, this can be done by designing another ASF centered in that area of motor cortex representing the specific limb. Further research has to show which body parts are most suited for this task. Finally, all work presented here has been done offline. Online versions of the ASF-algorithm are under development and will be presented in future work.

## References

[1] J.R. Wolpaw, N. Birbaumer, D.J. McFarland, G. Pfurtscheller, and T.M. Vaughan. Brain-computer interfaces for communication and control. *Clinical Neurophysiology*, 113(6):767–791, 2002.

[2] H.H. Jasper and W. Penfield. Electrocorticograms in man: effect of the voluntary movement upon the electrical activity of the precentral gyrus. *Arch. Psychiat. Z. Neurol.*, 183:163–174, 1949.

[3] G. Pfurtscheller, Ch. Neuper, D. Flotzinger, and M. Pregenzer. EEG-based discrimination between imagination of right and left hand movement. *Electroencephalography and Clinical Neurophysiology*, 103:642–651, 1997.

[4] G. Pfurtscheller and F.H. Lopes da Silva. Event-related EEG/MEG synchronization and desynchronization: basic principles. *Clinical Neurophysiology*, 110:1842–1857, 1999.

[5] G. Blanchard and B. Blankertz. BCI competition 2003 - data set IIa: Spatial patterns of self-controlled brain rythm modulations. *IEEE Transactions on Biomedical Engineering*, 51(6):1062–1066, 2004.

[6] Z.J. Koles. The quantitative extraction and topographic mapping of the abnormal components in the clinical EEG. *Electroencephalography and Clinical Neurophysiology*, 79:440–447, 1991.

[7] H. Ramoser, J. Mueller-Gerking, and G. Pfurtscheller. Optimal spatial filtering of single trial EEG during imagined hand movement. *IEEE Transactions on Rehabilitation Engineering*, 8(4):441–446, 2000.

[8] L.C. Parra, C.D. Spence, A.D. Gerson, and P. Sajda. Recipes for linear analysis of EEG. *Neuroimage*, 28:326–341, 2005.

[9] S. Baillet, J.C. Mosher, and R.M. Leahy. Electromagnetic brain mapping. *IEEE Signal Processing Magazine*, 18(6):14–30, 2001.

[10] P.L. Nunez and R. Shrinivasan. *Electric Fields of the Brain. The Neurophysics of EEG*. Oxford University Press, 2nd edition, 2006.

[11] B.N. Cuffin and D. Cohen. Comparison of the magnetoencephalogram and electroencephalogram. *Electroencephalography and Clinical Neurophysiology*, 47(2):132–146, 1979.

[12] A. Delorme and S. Makeig. EEGLAB: an open source toolbox for analysis of single-trial EEG dynamics. *Journal of Neuroscience Methods*, 134:9–21, 2004.

[13] R.O. Duda, P.E. Hart, and D.G. Stork. *Pattern Classification*. Wiley, 2nd edition, 2000.
